# Fisher-Optimal Neural Population Codes for High-Dimensional Diffeomorphic Stimulus Representations

**Zhuo Wang**
Department of Mathematics
University of Pennsylvania
Philadelphia, PA 19104
wangzhuo@sas.upenn.edu

**Alan A. Stocker**
Department of Psychology
University of Pennsylvania
Philadelphia, PA 19104
astocker@sas.upenn.edu

**Daniel D. Lee**
Department of Electrical and Systems Engineering
University of Pennsylvania
Philadelphia, PA 19104
ddlee@seas.upenn.edu

## Abstract

In many neural systems, information about stimulus variables is often represented in a distributed manner by means of a population code. It is generally assumed that the responses of the neural population are tuned to the stimulus statistics, and most prior work has investigated the optimal tuning characteristics of one or a small number of stimulus variables. In this work, we investigate the optimal tuning for diffeomorphic representations of high-dimensional stimuli. We analytically derive the solution that minimizes the $L_2$ reconstruction loss. We compared our solution with other well-known criteria such as maximal mutual information. Our solution suggests that the optimal weights do not necessarily decorrelate the inputs, and the optimal nonlinearity differs from the conventional equalization solution. Results illustrating these optimal representations are shown for some input distributions that may be relevant for understanding the coding of perceptual pathways.

## 1 Introduction

There has been much work investigating how information about stimulus variables is represented by a population of neurons in the brain [1]. Studies on motion perception [2, 3] and sound localization [4, 5] have demonstrated that these representations adapt to the stimulus statistics on various time scales [6, 7, 8, 9]. This raises the natural question of what encoding scheme is underlying this adaptive process?

To address this question, several assumptions about the neural representation and its overall objective need to be made. In the case of a one-dimensional stimulus, a number of theoretical approaches have previously been investigated. Some work have focused on the scenario with a single neuron [10, 11, 12, 13, 14, 15], while other work focused on the population level [16, 17, 18, 19, 20, 21, 22, 23], with different model and noise assumptions. However, the question becomes more difficult when considering adaptation to high dimensional stimuli. An interesting class of solutions to this question is related to independent component analysis (ICA) [24, 25, 26], which considers maximizing the amount of information in the encoding given a distribution of stimulus inputs. The use of mutual information as a metric to measure neural coding quality has also been discussed in [27].

In this paper, we study Fisher-optimal population codes for the diffeomorphic encoding of stimuli with multivariate Gaussian distributions. Using Fisher information, we investigate the properties of representations that would minimize the $L_2$ reconstruction error assuming an optimal decoder. The optimization problem is derived under a diffeomorphic assumption, i.e. the number of encoding neurons matches the dimensionality of the input and the nonlinearity is monotonic. In this case, the optimal solution can be found analytically and can be given a geometric interpretation. Qualitative differences between this solution and the previously studied information maximization solutions are demonstrated and discussed.

## 2 Model and Methods

### 2.1 Encoding and Decoding Model

We consider a $n$ dimensional stimulus input $\mathbf{s} = (s_1, \ldots, s_n)$ with prior distribution $p(\mathbf{s})$. In general, a population with $m$ neurons can have $m$ individual activation functions, $h_1(\mathbf{s}), \ldots, h_m(\mathbf{s})$ which determines the average firing rate of each neuron in response to the stimulus. However, the encoding process is affected by neural noise. Two commonly used models are Poisson noise model and constant Gaussian model, for which the observed firing rate vector $\mathbf{r} = (r_1, \ldots, r_m)$ follows the probabilistic distribution $p(\mathbf{r}|\mathbf{s})$, where

$$r_k T \sim \text{Poisson}(h_k(\mathbf{s})T) \qquad \text{(Poisson noise)} \qquad (1)$$
$$r_k T \sim \text{Gaussian}(h_k(\mathbf{s})T, VT) \qquad \text{(Gaussian noise)} \qquad (2)$$

As opposed to encoding, the decoding process involves constructing an estimator $\hat{\mathbf{s}}(\mathbf{r})$, which deterministically maps the response $\mathbf{r}$ to an estimate $\hat{\mathbf{s}}$ of the true stimulus $\mathbf{s}$. We choose a maximum likelihood estimator $\hat{s}_{\text{MLE}}(\mathbf{r}) = \arg\max_{\mathbf{s}} p(\mathbf{r}|\mathbf{s})$ because it simplifies the calculation due to its nice statistical properties as discussed in section 2.3.

### 2.2 Fisher Information Matrix

The Fisher information is a key concept widely used in optimal coding theory. For multiple dimensions, the Fisher information matrix is defined element-wise for each $\mathbf{s}$, as in [28],

$$\mathbf{I}_F(\mathbf{s})_{i,j} = \left\langle \frac{\partial}{\partial s_i} \log p(\mathbf{r}|\mathbf{s}) \cdot \frac{\partial}{\partial s_j} \log p(\mathbf{r}|\mathbf{s}) \,\middle|\, \mathbf{s} \right\rangle_{\mathbf{r}} \qquad (3)$$

In the supplementary section A we prove that the Fisher information matrix for a population of $m$ neurons is

$$\mathbf{I}_F(\mathbf{s}) = T \cdot \sum_{k=1}^{m} h_k(\mathbf{s})^{-1} \nabla h_k(\mathbf{s}) \cdot \nabla h_k(\mathbf{s})^T \qquad \text{(Poisson noise)} \qquad (4)$$

$$\boxed{\mathbf{I}_F(\mathbf{s}) = T \cdot \sum_{k=1}^{m} V^{-1} \nabla \tilde{h}_k(\mathbf{s}) \cdot \nabla \tilde{h}_k(\mathbf{s})^T} \qquad \text{(Gaussian noise)} \qquad (5)$$

where $T$ is length of the encoding time window and $V$ represents the variance of the constant Gaussian noise. The equivalence for two noise models can be established via the variance stabilizing transformation $\tilde{h}_k = 2\sqrt{h_k}$ [29]. Without loss of generality, throughout the paper we assume the Gaussian noise model for mathematical convenience. Also we will simply assume $V = 1, T = 1$ because they do not change the optimal solution for any Fisher information-related quantities.

### 2.3 Cramer-Rao Lower Bound

Ideally, a good neural population code should produce estimates $\hat{\mathbf{s}}$ that are close to the true value of the stimulus $\mathbf{s}$. However multiple measures exist for how well an estimate matches the true value. One possibility is the $L_2$ loss which is related to the Fisher information matrix via the Cramer-Rao lower bound [28]. For any unbiased estimator $\hat{\mathbf{s}}$, including the MLE,

$$\mathbf{cov}[\hat{\mathbf{s}} - \mathbf{s}] \geq \mathbf{I}_F(\mathbf{s})^{-1} \qquad (6)$$

in the sense that $\mathbf{cov}[\hat{\mathbf{s}} - \mathbf{s}] - \mathbf{I}_F(\mathbf{s})^{-1}$ is a positive semidefinite matrix. Being only a lower bound, the Cramer-Rao bound can be attained by the MLE $\hat{\mathbf{s}}$ because it is asymptotically efficient. The local $L_2$ decoding error $\langle \|\hat{\mathbf{s}} - \mathbf{s}\|^2 | \mathbf{s} \rangle_{\mathbf{r}} = \mathrm{tr}(\mathbf{cov}(\hat{\mathbf{s}} - \mathbf{s})) \geq \mathrm{tr}(\mathbf{I}_F(\mathbf{s})^{-1})$. In order to minimize the overall $L_2$ decoding error, one should minimize the attainable lower bound on the right side of Eq.(7), under appropriate constraints on $h_k(\cdot)$.

$$\left\langle \|\hat{\mathbf{s}} - \mathbf{s}\|^2 \right\rangle_{\mathbf{s}} \geq \left\langle \mathrm{tr}(\mathbf{I}_F(\mathbf{s})^{-1}) \right\rangle_{\mathbf{s}} \tag{7}$$

## 2.4 Mutual Information Limit

Another possible measurement of neural coding quality is the mutual information. This quantity does not explicitly rely on an estimator $\hat{\mathbf{s}}(\mathbf{r})$ but directly measures the mutual information between the response and the stimulus.

The link between mutual information and the Fisher information matrix was established in [16]. One goal (infomax) is to maximize the mutual information $I(\mathbf{r}, \mathbf{s}) = H(\mathbf{r}) - H(\mathbf{r}|\mathbf{s})$. Assuming perfect integration, the first term $H(\mathbf{r})$ asymptotically converges to a constant $H(\mathbf{s})$ for long encoding time because the noise is Gaussian. The second term $H(\mathbf{r}|\mathbf{s}) = \langle H(\mathbf{r}|\mathbf{s}^*) \rangle_{\mathbf{s}^*}$ because the noise is independent. For each $\mathbf{s}^*$, the conditional entropy $H(\mathbf{r}|\mathbf{s} = \mathbf{s}^*) \propto \frac{1}{2} \log \det \mathbf{I}_F(\mathbf{s}^*)$ since $\mathbf{r}|\mathbf{s}^*$ is asymptotically a Gaussian variable with covariance $\mathbf{I}_F(\mathbf{s}^*)$. Therefore the mutual information is

$$I(\mathbf{r}, \mathbf{s}) = const + \frac{1}{2} \langle \log \det \mathbf{I}_F(\mathbf{s}) \rangle_{\mathbf{s}} \tag{8}$$

## 2.5 Diffeomorphic Population

Before one can formalize the optimal coding problem, some assumptions about the neural population need to be made. Under a diffeomorphic assumption, the number of neurons ($m$) in the population matches the dimensionality ($n$) of the input stimulus. Each neuron projects the signal $\mathbf{s}$ onto its basis $\mathbf{w}_k$ and passes the one-dimensional projection $t_k = \mathbf{w}_k^T \mathbf{s}$ through a sigmoidal tuning curve $h_k(\cdot)$ which is bounded $0 \leq h_k(\cdot) \leq 1$. The tuning curve is

$$r_k = h_k(\mathbf{w}_k^T \mathbf{s}). \tag{9}$$

We would like to optimize for the nonlinear functions $h_1(\cdot), \ldots, h_n(\cdot)$ and the basis $\{\mathbf{w}_k\}_{k=1}^n$ simultaneously. We may assume $\|\mathbf{w}_k\| = 1$ since the scale can be compensated by the nonlinearity. Such an encoding scheme is called diffeomorphic because the population establishes a smooth and invertible mapping from the stimulus space $\mathbf{s} \in \mathbf{S}$ to the rate space $\mathbf{r} \in \mathbf{R}$. An arbitrary observation of the firing rate $\mathbf{r}$ can be first inverted to calculate the hidden variables $t_k = h_k^{-1}(r_k)$ and then linearly decoded to obtain $\hat{s}_{MLE}$.

Fig.1a shows how the encoding scheme is implemented by a neural network. Fig.1b illustrates explicitly how a 2D stimulus $\mathbf{s}$ is encoded by two neurons with basis $\mathbf{w}_1, \mathbf{w}_2$ and nonlinear mappings $h_1, h_2$.

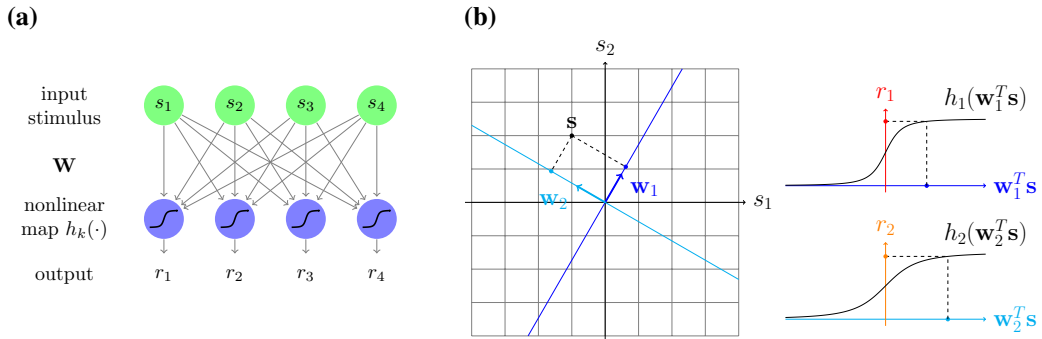

Figure 1: **(a)** Illustration of a neural network with diffeomorphic encoding. **(b)** The Linear-Nonlinear (LN) encoding process of 2D stimulus for a stimulus $\mathbf{s}$.

## 3 Review of One Dimensional Solution

In the case of encoding an one-dimensional stimulus, the diffeomorphic population is just one neuron with sigmoidal tuning curve $r = h(w \cdot s)$. The only two options $w = \pm 1$ is determined by whether the sigmoidal tuning curve is increasing or decreasing. Here we simply assume $w = 1$.

For the $L_2$-minimization problem, we want to minimize $\langle \mathrm{tr}(I_F(s)^{-1}) \rangle = \langle h'(s)^{-2} \rangle$ because of Eq.(5) and (7). Now apply Holder's inequality [30] to non-negative functions $p(s)/h'(s)^2$ and $h'(s)$,

$$
\underbrace{\left( \int \frac{p(s)}{h'(s)^2} \, ds \right)}_{\text{overall } L_2 \text{ loss}} \cdot \underbrace{\left( \int h'(s) \, ds \right)^2}_{=1} \geq \left( \int p(s)^{1/3} \, ds \right)^3 \tag{10}
$$

The minimum $L_2$ loss is attained by the optimal $h^*(s) \propto \int_{-\infty}^{s} p(t)^{1/3} dt$. For one dimensional Gaussian with variance $\mathbf{Var}[s]$, the right side of Eq.(10) is $6\sqrt{3}\pi \mathbf{Var}[s]$. This preliminary result will be useful for the high dimensional case discussed in Section 4 and 5.

On the other hand, for the infomax problem we want to maximize $I(\mathbf{r}, \mathbf{s})$ because of Eq.(5) and (8). Note that $\langle \log \det I_F(s) \rangle = 2\langle \log h'(s) \rangle$. By treating the sigmoidal activation function $h(s)$ as a cumulative probability distribution [10], we have

$$
\int p(s) \log h'(s) \, ds \leq \int p(s) \log p(s) \, ds \tag{11}
$$

because the KL-divergence $D_{KL}(p||h') = \int p(s) \log p(s) \, ds - \int p(s) \log h'(s) \, ds$ is non-negative. The optimal solution is $h^*(s) = \int_{-\infty}^{s} p(t) dt$ and the optimal value is $2H(p)$, where $H(p)$ is the differential entropy of the distribution $p(s)$. This $h^*(s)$ is exactly obtained by equalizing the output probability to maximize the entropy. For a one dimensional Gaussian with variance $\mathbf{Var}[s]$, the optimal value is $\log \mathbf{Var}[s] + const$.

## 4 Optimal Diffeomorphic Population

In the case of encoding high-dimensional random stimulus using a diffeomorphic population code, $n$ neurons encode $n$ stimulus dimensions. The gradient of the $k$-th neuron's tuning curve is $\nabla_k = h'_k(\mathbf{w}_k^T \mathbf{s})\mathbf{w}_k$ and the Fisher information matrix is thus

$$
\mathbf{I}_F(\mathbf{s}) = \sum_{k=1}^{n} \nabla_k \nabla_k^T = \sum_{k=1}^{n} h'_k(\mathbf{w}_k^T \mathbf{s})^2 \mathbf{w}_k \mathbf{w}_k^T = W H^2 W^T \tag{12}
$$

where $W = (\mathbf{w}_1, \ldots, \mathbf{w}_n)$ and $H = diag(h'_1(\mathbf{w}_1^T \mathbf{s}), \ldots, h'_n(\mathbf{w}_n^T \mathbf{s}))$. Using the fact that $\mathrm{tr}(AB) = \mathrm{tr}(BA)$ for any matrices $A, B$, we know $\mathrm{tr}(\mathbf{I}_F(s)^{-1}) = \mathrm{tr}((W^T)^{-1}H^{-2}W^{-1}) = \mathrm{tr}((W^T W)^{-1}H^{-2})$. Because $H^{-2}$ is diagonal, the $L_2$-min problem is simplified as

$$
\underset{\{\mathbf{w}_k, h_k(\cdot)\}, k=1\ldots n}{\text{minimize}} \quad L(W, H) = \langle \mathrm{tr}(\mathbf{I}_F(\mathbf{s})^{-1}) \rangle = \sum_{k=1}^{n} [(W^T W)^{-1}]_{kk} \int \frac{p(\mathbf{s})}{h'_k(\mathbf{w}_k^T \mathbf{s})^2} \, d\mathbf{s} \tag{13}
$$

If we define the marginal distribution

$$
p_k(t) = \int p(\mathbf{s}) \delta(t - \mathbf{w}_k^T \mathbf{s}) \, d\mathbf{s} \tag{14}
$$

then the optimization over $\mathbf{w}_k$ and $h_k$ can be decoupled in the following way. For any fixed $W$, the integral term can be evaluated by marginalizing out all those directions perpendicular to $\mathbf{w}_k$. As discussed in section 3, the optimal value $(\int p_k(t)^{1/3} \, dt)^3$ is attained when $h_k^{*\prime}(t) \propto p_k(t)^{1/3}$. The optimization problem is now

$$
\underset{\{\mathbf{w}_k\}, k=1\ldots n}{\text{minimize}} \quad L_{h^*}(W) = \sum_{k=1}^{n} [(W^T W)^{-1}]_{kk} \left( \int p_k(t)^{1/3} \, dt \right)^3 \tag{15}
$$

In general, analytically optimizing such a term for arbitrary prior distribution $p(\mathbf{s})$ is intractable. However if $p(\mathbf{s})$ is multivariate Gaussian then the optimization can be further simplified and solved analytically, as discussed in the following section.

# 5 Stimulus with Gaussian Prior

We consider the case when the stimulus prior is Gaussian $\mathcal{N}(\mathbf{0}, \Sigma)$. This assumption allows us to calculate the marginal distribution along any direction $\mathbf{w}_k$ as an one-dimensional Gaussian with mean zero and variance $\mathbf{w}_k^T \Sigma \mathbf{w}_k = (W^T \Sigma W)_{kk}$. By plugging in the Gaussian density $p_k(t)$ and using the fact we derived in Section 3, we can further simplify the $L_2$-optimization problem as

$$\underset{\{\mathbf{w}_k\}, k=1...n}{\text{minimize}} \quad L_{h^*}(W) = 6\sqrt{3}\pi \cdot \sum_{k=1}^{n} [(W^T W)^{-1}]_{kk} (W^T \Sigma W)_{kk} \tag{16}$$

## 5.1 Geometric Interpretation

In the above optimization problem, $(W^T \Sigma W)_{kk}$ has a clear and simple meaning – it is the variance of the marginal distribution $p_k(t)$. For term $[(W^T W)^{-1}]_{kk}$, notice that $W^T W$ is the inner product matrix of the basis $\{\mathbf{w}_k\}_{k=1}^{n}$, i.e. $(W^T W)_{ij} = \mathbf{w}_i^T \mathbf{w}_j$. Using the adjoint method we can calculate the diagonal elements of $(W^T W)^{-1}$,

$$[(W^T W)^{-1}]_{kk} = \frac{\det(W_k^T W_k)}{\det(W^T W)} \tag{17}$$

where $W_k^T W_k$ is the inner product matrix of leave-$\mathbf{w}_k$-out basis $\{\mathbf{w}_1, \ldots, \mathbf{w}_{k-1}, \mathbf{w}_{k+1}, \ldots, \mathbf{w}_n\}$. Let $\theta_k$ be the angle between $\mathbf{w}_k$ and the hyperplane spanned by all other basis vectors (see Fig.2). The diagonal element is just $[(W^T W)^{-1}]_{kk} = (\det W_k / \det W)^2 = (\sin \theta_k)^{-2}$ simply because

$$\underbrace{\text{Volume} (\{\mathbf{w}_1, \ldots, \mathbf{w}_n\})}_{n \text{ dim parallelogram}} = \underbrace{\text{Volume} (\{\mathbf{w}_1, \ldots, \mathbf{w}_{k-1}, \mathbf{w}_{k+1}, \ldots, \mathbf{w}_n\})}_{n-1 \text{ dim base parallelogram}} \cdot \underbrace{|\mathbf{w}_k| \cdot \sin \theta_k}_{\text{height}}, \tag{18}$$

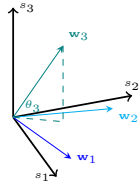

Figure 2: Illustration of $\theta_k$. In this example, $\mathbf{w}_1$ and $\mathbf{w}_2$ are on the $s_1$-$s_2$ plane. $\theta_3$ is just the angle between $\mathbf{w}_3$ and its projection on the $s_1$-$s_2$ plane.

The optimization involves two competing parts. Minimizing $(W^T \Sigma W)_{kk}$ makes all those directions with small variance favorable. Meanwhile, minimizing $[(W^T W)^{-1}]_{kk} = (\sin \theta_k)^{-2}$ strongly penalizes neurons having similar tuning directions with the rest of population. To qualitatively summarize, the optimal population would tend to encode those directions with small variance while keeping certain degree of population diversity.

## 5.2 General Solution

Due to space limitations, we will only present the optimal solution here and the derivation can be found in Appendix C in the supplementary notes. For any covariance matrix $\Sigma$, the optimal solution for Eq.(16) is

$$W^* = \Sigma^{-1/4} U, \quad \text{where } U^T U = I \text{ and } (U^T \Sigma^{1/2} U)_{kk} = \frac{1}{n} \operatorname{tr}(\Sigma^{1/2}) \text{ for all } k = 1, \ldots, n \tag{19}$$

Such unitary matrix $U$ is guaranteed to exist yet may not be unique. See Appendix D for a detailed discussion. In general for dimension $n$, the solution has a manifold structure with dimension not less than $(n-1)(n-2)/2$. For $n = 2$ the solution can be easily derived. Let $\Sigma = diag(\sigma_x^2, \sigma_y^2)$. Then optimal solution is given by

$$U = \frac{1}{\sqrt{2}} \begin{pmatrix} 1 & -1 \\ 1 & 1 \end{pmatrix}, \quad W_{\text{L2}}^* = \Sigma^{-1/4} U = \frac{1}{\sqrt{2}} \begin{pmatrix} \frac{1}{\sqrt{\sigma_x}} & -\frac{1}{\sqrt{\sigma_x}} \\ \frac{1}{\sqrt{\sigma_y}} & \frac{1}{\sqrt{\sigma_y}} \end{pmatrix} \tag{20}$$

This 2D solution is special and is unique under reflection and permutation unless the prior distribution is spherically symmetric i.e. $\Sigma = aI$.

# 6 Comparison with Infomax Solution

Previous studies have focused on finding solutions that maximize the mutual information (infomax) between the stimulus and the neural population response. This is related to independent component analysis (ICA) [24]. Mutual information can be maximized if and only if each neuron encodes an independent component of the stimulus and uses the proper nonlinear tuning curve. Ideally, the joint distribution $p(\mathbf{s})$ can be decomposed as the product of $n$ one dimensional components $\prod_{k=1}^{n} p_k(W_k(\mathbf{s}))$. For a Gaussian prior with covariance $\Sigma$, the infomax solution is

$$W_{\text{info}}^* = \Sigma^{-1/2} U \quad \Rightarrow \quad \mathbf{cov}(W_{\text{info}}^{*T}\mathbf{s}) = U^T \Sigma^{-1/2} \cdot \Sigma \cdot \Sigma^{-1/2} U = I \tag{21}$$

where $\Sigma^{-1/2}$ is the whitening matrix and $U$ is an arbitrary unitary matrix. The derivation can be found in Appendix E. In the same 2D example where $\Sigma = diag(\sigma_x^2, \sigma_y^2)$, the family of optimal solutions is parametrized by an angular variable $\phi$

$$U(\phi) = \frac{1}{\sqrt{2}} \begin{pmatrix} \cos\phi & -\sin\phi \\ \sin\phi & \cos\phi \end{pmatrix}, \quad W_{\text{info}}^*(\phi) = \Sigma^{-1/2} \cdot U(\phi) = \begin{pmatrix} \frac{\cos\phi}{\sigma_x} & -\frac{\sin\phi}{\sigma_x} \\ \frac{\sin\phi}{\sigma_y} & \frac{\cos\phi}{\sigma_y} \end{pmatrix} \tag{22}$$

In Fig.3 we compare $W_{\text{info}}^*(\phi)$ and $W_{\text{L2}}^*$ for different prior covariances. One observation is that, $L_2$ optimal neurons do not fully decorrelate input signals unless the Gaussian prior is spherical. By correlating the input signal and encoding redundant information, the channel signal to noise ratio (SNR) can be balanced to reduce the vulnerability of those independent channels with low SNR. As a consequence, the overall $L_2$ performance is improved at the cost of transferring a suboptimal amount of information. Another important observation is that the infomax solution allows a greater degree of symmetry – Eq.(21) holds for arbitrary unitary matrices while Eq.(19) holds only for a subset of them.

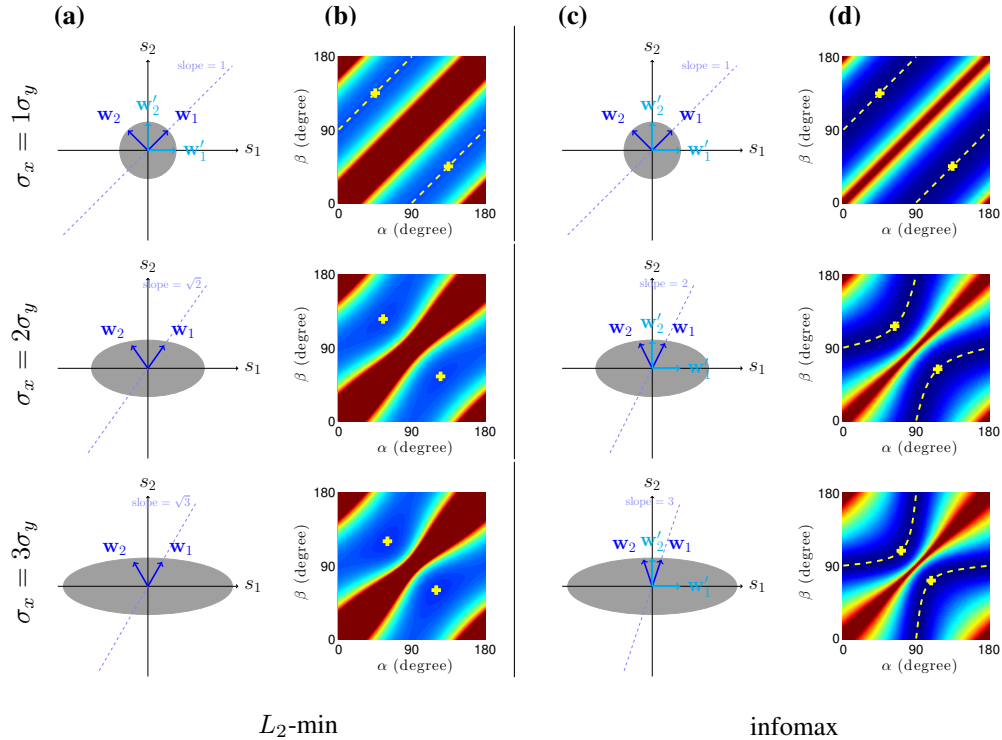

Figure 3: Comparison of $L_2$-min and infomax optimal solution for 2D case. Each row represents the result for different ratio $\sigma_x/\sigma_y$ for the prior distribution. **(a)** The optimal pair of basis vectors $\mathbf{w}_1, \mathbf{w}_2$ for $L_2$-min with the prior covariance ellipse is unique unless the prior distribution has rotational symmetry. **(b)** The loss function with "+" marking the optimal solution shown in (a). **(c)** One pair of optimal basis vector $\mathbf{w}_1, \mathbf{w}_2$ for infomax with the prior covariance ellipse. **(d)** The loss function with "+" marking the optimal solution shown in (c).

# 7 Application – 16-by-16 Gaussian Images

In this section we apply our diffeomorphic coding scheme to an image representation problem. We assume that the intensity values of all pixels from a set of 16-by-16 images follow a 256-D Gaussian distribution. Instead of directly defining the pairwise covariance between pixels of $\mathbf{s}$, we calculate its real Fourier components $\hat{\mathbf{s}}$

$$\tilde{\mathbf{s}} = F^T \mathbf{s} \quad \Leftrightarrow \quad \mathbf{s} = F\hat{\mathbf{s}} \tag{23}$$

where the real Fourier matrix is $F = (f_1, \ldots, f_{256})$ with each filter $f_a$ and its spatial frequency $\vec{k}_a$. The covariance of those Fourier components $\tilde{\mathbf{s}}$ is typically assumed to be diagonal and the power decays following some power law

$$\mathbf{cov}(\tilde{\mathbf{s}}) = D = diag(\sigma_1^2, \ldots, \sigma_n^2), \quad \text{where } \sigma_a^2 \propto |\vec{k}_a|^{-\beta}, \quad \beta > 0 \tag{24}$$

Therefore the original stimulus $\mathbf{s}$ has covariance $\mathbf{cov}(\mathbf{s}) = \Sigma = FDF^T$. Such image statistics are called *stationary* because the covariance between pair of pixels is fully determined by their relative position. For the stimulus $s$ with covariance $\Sigma$, one naive choice of $L_2$ optimal filter is simply

$$W_{\text{L2}}^* = \Sigma^{-1/4} \cdot I = FD^{-1/4}F^T \tag{25}$$

because $\Sigma^{1/2} = FD^{1/2}F^T$ has constant diagonal terms (See Appendix F for detailed calculation) and $U = I$ qualifies for Eq.(19). The covariance matrix and one sample image generated from $\Sigma$ is plotted in Fig. 4(a)-(c) below.

**(a)**            **(b)**            **(c)**

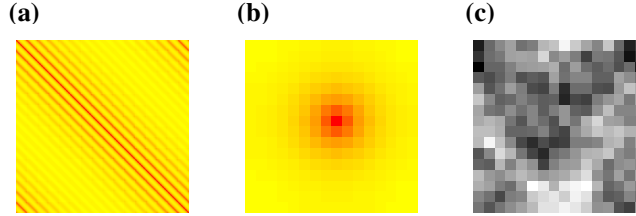

Figure 4: For $\beta = 2.5$ in the power law: **(a)** The $256 \times 256$ covariance matrix $\Sigma$. **(b)** One column of $\Sigma$ reshaped to $16 \times 16$ matrix representing the covariance between any pixels and a fixed pixel in the center. **(c)** A random sample from the Gaussian distribution with covariance $\Sigma$.

In addition, we have numerically computed the $L_2$ loss using a family of filters

$$W_\gamma = FD^{-\gamma}F^T, \quad \gamma \in [0, 1/2] \tag{26}$$

Note that when $\gamma = 0$, we have the naive filter $W_0 = FF^T = I$ which does nothing to the input stimulus; when $\gamma = 1/4$ or $1/2$, we revisit the $L_2$ optimal filter or the infomax filter, respectively. As we can see from Fig. 5(a)-(d), the $L_2$ optimal filter half-decorrelates the input stimulus channels to keep the balance between the simplicity of the filters and the simplicity of the correlation structure.

In each simulation run, a set of 10,000 16-by-16 images is randomly sampled from the multivariate Gaussian distribution with zero mean and covariance matrix $\Sigma$. For each stimulus image $\mathbf{s}$, we calculate $\mathbf{y} = W_\gamma^T \mathbf{s}$ and $z_k = h_k(y_k) + \eta_k$ to simulate the encoding process. Here $h_k(y) \propto \int_{-\infty}^{y} p_k(t)^{1/3} dt$ and $p_k(t)$ is Gaussian $\mathcal{N}(0, (W_\gamma^T \Sigma W_\gamma)_{kk})$. The additive Gaussian noise $\eta_k$ is independent Gaussian $\mathcal{N}(0, 10^{-4})$. To decode, we just calculate $\hat{y}_k = h_k^{-1}(z_k)$ and $\hat{\mathbf{s}} = (W_\gamma^T)^{-1}\hat{\mathbf{y}}$. Then we measure the $L_2$ loss $\|\hat{\mathbf{s}} - \mathbf{s}\|_2^2$. This procedure is repeated 20 times and the result is plotted in Fig. 5(e).

# 8 Discussion and Conclusions

In this paper, we have studied the an optimal diffeomorphic neural population code which minimizes the $L_2$ reconstruction error. The population of neurons is assumed to have sigmoidal activation functions encoding linear combinations of a high dimensional stimulus with a multivariate Gaussian

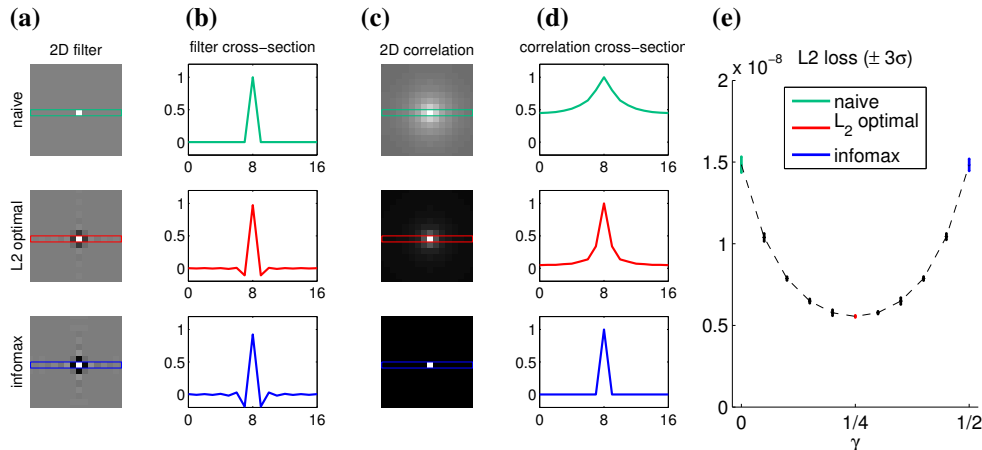

Figure 5: **(a)** The 2D filter $W_\gamma$ of one specific neuron for $\gamma = 0, 1/4, 1/2$ from top to bottom. **(b)** The cross-section of the filter $W_\gamma$ on one specific row boxed in (a), plotted as a function. **(c)** The correlation of the 2D filtered stimulus, between one specific neuron and all neurons. **(d)** The cross-section of the 2D correlation of the filtered stimulus, between the neuron and other neurons on the same row. **(e)** The simulation result of $L_2$ loss for different filter $W_\gamma$ and optimal nonlinearity $h$ and the vertical bar shows the $\pm 3\sigma$ interval across trials.

distribution. The optimal solution is provided and compared with solutions which maximize the mutual information.

In order to derive the optimal solution, we first show that the Poisson noise model is equivalent to the constant Gaussian noise under the variance stabilizing transformation. Then we relate the $L_2$ reconstruction error to the trace of inverse Fisher information matrix via the Cramer-Rao bound. Minimizing this bound leads to the global optimal solution in the asymptotic limit of long integration time. The general $L_2$-minimization problem can be simplified and the optimal solution is analytically derived when the stimulus distribution is Gaussian.

Compared to the infomax solutions, a careful evaluation and calculation of the Fisher information matrix is needed for $L_2$ minimization. The manifold of $L_2$ optimal solutions possess a lower dimensional structure compared to the infomax solution. Instead of decorrelating the input statistics, the $L_2$-min solution maintains a certain degree of correlation across the channels. Our result suggests that maximizing mutual information and minimizing the overall decoding loss are not the same in general – encoding redundant information can be beneficial to improve reconstruction accuracy. This principle may explain the existence of correlations at many layers in biological perception systems.

As an example, we have applied our theory to 16-by-16 images with stationary pixel statistics. The optimal solution exhibits center-surround receptive fields, but with a decay differing from those found by decorrelating solutions. We speculate that these solutions may better explain observed correlations measured in certain neural areas of the brain. Finally, we acknowledge the support of the Office of Naval Research.

# References

[1] K Kang, RM Shapley, and H Sompolinsky. Information tuning of populations of neurons in primary visual cortex. *Journal of neuroscience*, 24(15):3726–3735, 2004.

[2] AP Georgopoulos, AB Schwartz, and RE Kettner. Adaptation of the motion-sensitive neuron h1 is generated locally and governed by contrast frequency. *Science*, 233:1416–1419, 1986.

[3] FE Theunissen and JP Miller. Representation of sensory information in the cricket cercal sensory system. II. information theoretic calculation of system accuracy and optimal tuning-curve widths of four primary interneurons. *J Neurophysiol*, 66(5):1690–1703, November 1991.

[4] DC Fitzpatrick, R Batra, TR Stanford, and S Kuwada. A neuronal population code for sound localization. *Nature*, 388:871–874, 1997.

[5] NS Harper and D McAlpine. Optimal neural population coding of an auditory spatial cue. *Nature*, 430:682–686, 2004.

[6] N Brenner, W Bialek, and R de Ruyter van Steveninck. Adaptive rescaling maximizes information transmission. *Neuron*, 26:695–702, 2000.

[7] Tvd Twer and DIA MacLeod. Optimal nonlinear codes for the perception of natural colours. *Network: Computation in Neural Systems*, 12(3):395–407, 2001.

[8] I Dean, NS Harper, and D McAlpine. Neural population coding of sound level adapts to stimulus statistics. *Nature neuroscience*, 8:1684–1689, 2005.

[9] Y Ozuysal and SA Baccus. Linking the computational structure of variance adaptation to biophysical mechanisms. *Neuron*, 73:1002–1015, 2012.

[10] SB Laughlin. A simple coding procedure enhances a neurons information capacity. *Z. Naturforschung*, 36c(3):910–912, 1981.

[11] J-P Nadal and N Parga. Non linear neurons in the low noise limit: A factorial code maximizes information transfer, 1994.

[12] M Bethge, D Rotermund, and K Pawelzik. Optimal short-term population coding: when Fisher information fails. *Neural Computation*, 14:2317–2351, 2002.

[13] M Bethge, D Rotermund, and K Pawelzik. Optimal neural rate coding leads to bimodal firing rate distributions. *Netw. Comput. Neural Syst.*, 14:303–319, 2003.

[14] MD McDonnell and NG Stocks. Maximally informative stimuli and tuning curves for sigmoidal rate-coding neurons and populations. *Phys. Rev. Lett.*, 101:058103, 2008.

[15] Z Wang, A Stocker, and DD Lee. Optimal neural tuning curves for arbitrary stimulus distributions: Discrimax, infomax and minimum lp loss. *Adv. Neural Information Processing Systems*, 25:2177–2185, 2012.

[16] N Brunel and J-P Nadal. Mutual information, fisher information and population coding. *Neural Computation*, 10(7):1731–1757, 1998.

[17] K Zhang and TJ Sejnowski. Neuronal tuning: To sharpen or broaden? *Neural Computation*, 11:75–84, 1999.

[18] A Pouget, S Deneve, J-C Ducom, and PE Latham. Narrow versus wide tuning curves: Whats best for a population code? *Neural Computation*, 11:85–90, 1999.

[19] H Sompolinsky and H Yoon. The effect of correlations on the fisher information of population codes. *Advances in Neural Information Processing Systems*, 11, 1999.

[20] AP Nikitin, NG Stocks, RP Morse, and MD McDonnell. Neural population coding is optimized by discrete tuning curves. *Phys. Rev. Lett.*, 103:138101, 2009.

[21] D Ganguli and EP Simoncelli. Implicit encoding of prior probabilities in optimal neural populations. *Adv. Neural Information Processing Systems*, 23:658–666, 2010.

[22] S Yaeli and R Meir. Error-based analysis of optimal tuning functions explains phenomena observed in sensory neurons. *Front Comput Neurosci*, 4, 2010.

[23] E Doi and MS Lewicki. Characterization of minimum error linear coding with sensory and neural noise. *Neural Computation*, 23, 2011.

[24] AJ Bell and TJ Sejnowski. An information-maximization approach to blind separation and blind deconvolution. *Neural Computation*, 7:1129–1159, 1995.

[25] DJ Field BA Olshausen. Emergence of simple-cell receptive field properties by learning a sparse code for natural images. *Nature*, 381:607–609, 1996.

[26] A Hyvarinen and E Oja. Independent component analysis: Algorithms and applications. *Neural Networks*, 13:411–430, 2000.

[27] P Berens, A Ecker, S Gerwinn, AS Tolias, and M Bethge. Reassessing optimal neural population codes with neurometric functions. *Proceedings of the National Academy of Sciences*, 11:4423–4428, 2011.

[28] TM Cover and J Thomas. *Elements of Information Theory*. Wiley, 1991.

[29] EL Lehmann and G Casella. *Theory of point estimation*. New York: Springer-Verlag., 1999.

[30] GH Hardy, JE Littlewood, and G Polya. *Inequalities, 2nd ed*. Cambridge University Press, 1988.

